# Improvisation and Learning

**Judy A. Franklin**[*]
Computer Science Department
Smith College
Northampton, MA 01063
*jfranklin@cs.smith.edu*

## Abstract

This article presents a 2-phase computational learning model and application. As a demonstration, a system has been built, called CHIME for Computer Human Interacting Musical Entity. In phase 1 of training, recurrent back-propagation trains the machine to reproduce 3 jazz melodies. The recurrent network is expanded and is further trained in phase 2 with a reinforcement learning algorithm and a critique produced by a set of basic rules for jazz improvisation. After each phase CHIME can interactively improvise with a human in real time.

## 1 Foundations

Jazz improvisation is the creation of a jazz melody in real time. Charlie Parker, Dizzy Gillespie, Miles Davis, John Coltrane, Charles Mingus, Thelonious Monk, and Sonny Rollins et al. were the founders of bebop and post bop jazz [9] where drummers, bassists, and pianists keep the beat and maintain harmonic structure. Other players improvise over this structure and even take turns improvising for 4 bars at a time. This is called trading fours.

Meanwhile, artificial neural networks have been used in computer music [4, 12]. In particular, the work of (Todd [11]) is the basis for phase 1 of CHIME, a novice machine improvisor that learns to trade fours. Firstly, a recurrent network is trained with back-propagation to play three jazz melodies by Sonny Rollins [1], as described in Section 2. Phase 2 uses actor-critic reinforcement learning and is described in Section 3. This section is on jazz basics.

### 1.1 Basics: Chords, the ii-V-I Chord Progression and Scales

The harmonic structure mentioned above is a series of chords that may be reprated and that are often grouped into standard subsequences. A chord is a group of notes played simultaneously. In the chromatic scale, C-Db-D-Eb-E-F-Gb-G-Ab-A-Bb-B-C, notes are separated by a half step. A flat (b) note is a half step below the original note; a sharp (#) is a half above. Two half steps are a whole step. Two whole steps are a major third. Three half steps are a minor third.

A major triad (chord) is the first or tonic note, then the note a major third up, then the note a minor third up. When F is the tonic, F major triad is F-A-C. A minor triad (chord) is the tonic

---

[*]www.cs.smith.edu/~jfrankli

then a minor third, then a major third. F minor triad is F-Ab-C. The diminished triad is the tonic, then a minor third, then a minor third. F diminished triad is F-Ab-Cb. An augmented triad is the tonic, then a major third, then a major third. The F augmented triad is F-A-Db.

A third added to the top of a triad forms a seventh chord. A major triad plus a major third is the major seventh chord. F-A-C-E is the F major seventh chord (Fmaj7). A minor triad plus a minor third is a minor seventh chord. For F it is F-Ab-C-Eb (Fm7). A major triad plus a minor third is a dominant seventh chord. For F it is F-A-C-Eb (F7). These three types of chords are used heavily in jazz harmony. Notice that each note in the chromatic scales can be the tonic note for any of these types of chords.

A scale, a subset of the chromatic scale, is characterized by note intervals. Let W be a whole step and H be a half. The chromatic scale is HHHHHHHHHHHH. The major scale or ionian mode is WWHWWWH. F major scale is F-G-A-Bb-C-D-E-F. The notes in a scale are degrees; E is the seventh degree of F major. The first, third, fifth, and seventh notes of a major scale are the major seventh chord. The first, third, fifth, and seventh notes of other modes produce the minor seventh and dominant seventh chords. Roman numerals represent scale degrees and their seventh chords. Upper case implies major or dominant seventh and lower case implies minor seventh [9]. The major seventh chord starting at the scale tonic is the I (one) chord. G is the second degree of F major, and G-Bb-D-F is Gm7, the ii chord, with respect to F. The ii-V-I progression is prevalent in jazz [9], and for F it is Gm7-C7-Fmaj7. The minor ii-V-i progression is obtained using diminished and augmented triads, their seventh chords, and the aeolian mode. Seventh chords can be extended by adding major or minor thirds, e.g. Fmaj9, Fmaj11, Fmaj13, Gm9, Gm11, and Gm13. Any extension can be raised or lowered by 1 step [9] to obtain, e.g. Fmaj7#11, C7#9, C7b9, C7#11.

Most jazz compositions are either the 12 bar blues or sectional forms (e.g. ABAB, ABAC, or AABA) [8]. The 3 Rollins songs are 12 bar blues. "Blue 7" has a simple blues form. In "Solid" and "Tenor Madness", Rollins adds bebop variations to the blues form [1]. ii-V-I and VI-II-V-I progressions are added and G7+9 substitutes for the VI and F7+9 for the V (see section 1.2 below); the II-V in the last bar provides the turnaround to the I of the first bar to foster smooth repetition of the form. The result is at left and in Roman numeral notation at right:

| Bb7 | Bb7 | Bb7 | Bb7 |
|-----|-----|-----|-----|
| Eb7 | Eb7 | Bb7 | G7+9 |
| Cm7 | F7 | Bb7 G7+9 | C7 F7+9 |

| I | I | I | I |
|---|---|---|---|
| IV | IV | I | VI |
| ii | V | I VI | II V |

## 1.2 Scale Substitutions and Rules for Reinforcement Learning

First note that the theory and rules derived in this subsection are used in Phase 2, to be described in Section 3. They are presented here since they derive from the jazz basics immediately preceding. One way a novice improvisor can play is to associate one scale with each chord and choose notes from that scale when the chord is presented in the musical score. Therefore, Rule 1 is that an improvisor may choose notes from a "standard" scale associated with a chord. Next, the 4th degree of the scale is often avoided on a major or dominant seventh chord (Rule 3), unless the player can resolve its dissonance. The major 7th is an avoid note on a dominant seventh chord (Rule 4) since a dominant seventh chord and its scale contain the flat 7th, not the major 7th.

Rule 2 contains many notes that can be added. A brief rationale is given next. The C7 in Gm7-C7-Fmaj7 may be replaced by a C7#11, a C7+ chord, or a C7b9b5 or C7alt chord [9]. The scales for C7+ and C7#11 make available the raised fourth (flat 5), and flat 6 (flat 13) for improvising. The C7b9b5 and C7alt (C7+9) chords and their scales make available the flat9, raised 9, flat5 and raised 5 [1]. These substitutions provide the notes of Rule 2. These rules (used in phase 2) are stated below, using for reinforcement values very bad (-1.0), bad (-0.5), a little bad (-0.25), ok (0.25), good (0.5), and very good (1.0). The rules are discussed

further in Section 4.

**The Rule Set:**

1) Any note in the scale associated with the chord is ok (except as noted in rule 3).

2) On a dominant seventh, hip notes 9, flat9, #9, #11, 13 or flat13 are very good. One hip note 2 times in a row is a little bad. 2 hip notes more than 2 times in a row is a little bad.

3) If the chord is a dominant seventh chord, a natural 4th note is bad.

4) If the chord is a dominant seventh chord, a natural 7th is very bad.

5) A rest is good unless it is held for more than 2 16th notes and then it is very bad.

6) Any note played longer than 1 beat (4 16th notes) is very bad.

7) If two consecutive notes match the human's, that is good.

# 2 CHIME Phase 1

In Phase 1, supervised learning is used to train a recurrent network to reproduce the three Sonny Rollins melodies.

## 2.1 Network Details and Training

The recurrent network's output units are linear. The hidden units are nonlinear (logistic function). Todd [11] used a Jordan recurrent network [6] for classical melody learning and generation. In CHIME, a Jordan net is also used, with the addition of the chord as input (Figure 1. 24 of the 26 outputs are notes (2 chromatic octaves), the 25th is a rest, and the 26th indicates a new note. The output with the highest value above a threshold is the next note, including the rest output. The new note output indicates if this is a new note, or if it is the same note being held for another time step ($16^{th}$ note resolution).

The 12 chord inputs (12 notes in a chromatic scale), are 1 or 0. A chord is represented as its first, third, fifth, and seventh notes and it "wraps around" within the 12 inputs. E.g., the Fm7 chord F-Ab-C-Eb is represented as C, Eb, F, Ab or 100101001000. One plan input per song enables distinguishing between songs. The 26 context inputs use eligibility traces, giving the hidden units a decaying history of notes played. CHIME (as did Todd) uses teacher forcing [13], wherein the target outputs for the previous step are used as inputs (so erroneous outputs are not used as inputs). Todd used from 8 to 15 hidden units; CHIME uses 50. The learning rate is 0.075 (Todd used 0.05). The eligibility rate is 0.9 (Todd used 0.8). Differences in values perhaps reflect contrasting styles of the songs and available computing power.

Todd used 15 output units and assumed a rest when all note units are "turned off." CHIME uses 24 output note units (2 octaves). Long rests in the Rollins tunes require a dedicated output unit for a rest. Without it, the note outputs learned to turn off all the time. Below are results of four representative experiments. In all experiments, 15,000 presentations of the songs were made. Each song has 192 16th note events. All songs are played at a fixed tempo. Weights are initialized to small random values. The squared error is the average squared error over one complete presentation of the song. "Finessing" the network may improve these values. The songs are easily recognized however, and an exact match could impair the network's ability to improvise. Figure 2 shows the results for "Solid."

**Experiment 1**. Song: Blue Seven. Squared error starts at 185, decreases to 2.67.

**Experiment 2**. Song: Tenor Madness. Squared error starts at 218, decreases to 1.05.

**Experiment 3**. Song: Solid. Squared error starts at 184, decreases to 3.77.

**Experiment 4**. Song: All three songs: Squared error starts at 185, decreases to 36.

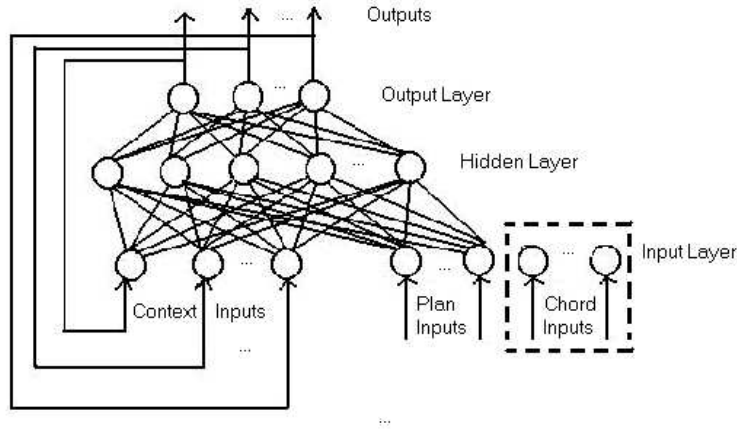

Figure 1: Jordan recurrent net with addition of chord input

## 2.2  Phase 1 Human Computer Interaction in Real Time

In trading fours with the trained network, human note events are brought in via the MIDI interface [7]. Four bars of human notes are recorded then given, one note event at a time to the context inputs (replacing the recurrent inputs). The plan inputs are all 1. The chord inputs follow the "Solid" form. The machine generates its four bars and they are played in real time. Then the human plays again, etc. An accompaniment (drums, bass, and piano), produced by Band-in-a-Box software (PG Music), keeps the beat and provides chords for the human.

Figure 3 shows an interaction. The machine's improvisations are in the second and fourth lines. In bar 5 the flat 9 of the Eb7 appears; the E. This note is used on the Eb7 and Bb7 chords by Rollins in "Blue 7", as a "passing tone." D is played in bar 5 on the Eb7. D is the natural 7 over Eb7 (with its flat 7) but is a note that Rollins uses heavily in all three songs, and once over the Eb7. It may be a response to the rest and the Bb played by the human in bar 1. D follows both a rest and a Bb in many places in "Tenor Madness" and "Solid." In bar 6, the long G and the Ab (the third then fourth of Eb7) figure prominently in "Solid." At the beginning of bar 7 is the 2-note sequence Ab-E that appears in exactly the same place in the song "Blue 7." The focus of bars 7 and 8 is jumping between the 3rd and 4th of Bb7. At the end of bar 8 the machine plays the flat 9 (Ab) then the flat 3 (Bb), of G7+9. In bars 13-16 the tones are longer, as are the human's in bars 9-12. The tones are the 5th, the root, the 3rd, the root, the flat 7, the 3rd, the 7th, and the raised fourth. Except for the last 2, these are chord tones.

## 3  CHIME Phase 2

In Phase 2, the network is expanded and trained by reinforcement learning to improvise according to the rules of Section 1.2 and using its knowledge of the Sonny Rollins songs.

### 3.1  The Expanded Network

Figure 4 shows the phase 2 network. The same inputs plus 26 human inputs brings the total to 68. The weights obtained in phase 1 initialize this network. The plan and chord weights

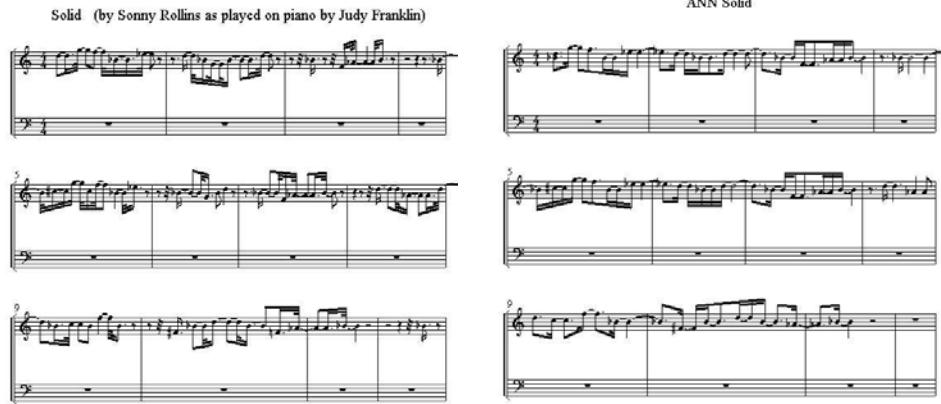

Figure 2: At left "Solid" played by a human; at right the song reproduced by the ANN.

are the same. The weights connecting context units to the hidden layer are halved. The same weights, halved, connect the 26 human inputs to the hidden layer. Each output unit gets the 100 hidden units' outputs as input. The original 50 weights are halved and used as initial values of the two sets of 50 hidden unit weights to the output unit.

## 3.2  SSR and Critic Algorithms

Using actor-critic reinforcement learning ([2, 10, 13]), the actor chooses the next note to play. The critic receives a "raw" reinforcement signal $r$ from the critique made by the rules of Section 1.2. For output j, the SSR (actor) computes mean $\mu_j = \sum_{i=0}^{n} w_{ij}x_i$. A Gaussian distribution with mean $\mu_j$ and standard deviation $\sigma_j$ chooses the output $y_j$. $r$ is generated, the critic modifies $r$ and produces $r_c$. $r_c$ is further modified by a self-scaling algorithm that tracks, via moving average, the maximum and minimum reinforcement and uses them to scale the signal to produce $\hat{r} = exp(r_c - rmax) - exp(rmin - r_c)$.

$$rmax(t+1) = max\{rmax(t), r_c\} \qquad rmin(t+1) = min\{rmin(t), r_c\}$$
$$rmax(t+1) = \lambda rmax(t+1) + (1-\lambda)r_c \quad rmin(t+1) = \lambda rmin(t+1) + (1-\lambda)r_c$$

The goal is to make small gains in reinforcement more noticeable and to scale the values between -1 and 1. If $r_c = rmax$, then $\hat{r} = 1 - exp(rmin - rmax) = 1 - \epsilon$ and if $r_c = rmin$, then $\hat{r} = exp(rmin - rmax) - 1 = \epsilon - 1$. If $\epsilon << 0$, the extremes of -1 and 1 are approached. The weight and standard deviation updates use $\hat{r}$:

$$w_{ij}(t+1) = w_{ij}(t) + \alpha\hat{r}(y_j - \mu_j)\partial\mu_j/\partial w_{ij}$$
$$\sigma_j(t+1) = max\{\sigma_{max}, min\{\gamma\sigma_j(t) + (1-\gamma)*(rmax - rmin), \sigma_{min}\}\}, 0 < \gamma < 1$$

If the difference between the max and min reinforcement stays large, over time $\sigma_j$ will increase (to a max of $\sigma_{max}$) and allow more exploration. When rmax-rmin is small, over time $\sigma_j$ will shrink (to a min of $\sigma_{min}$). The actor's hidden units are updated using backpropagation as before, using $\hat{r}*(y_j - \mu_j)$ as "error." See [3, 5, 13] for more details on the SSR algorithm and its precursors.

The critic inputs are the outputs of the hidden layer of the actor network; it "piggy-backs" on the actor and uses its learned features (see Figure 5). This also alleviates the computational burden so it can run in real time. There are delays in reward, e.g. in that a note played too many times in a row may result in punishment, and if 2 notes in a row coincide with the human's it is rewarded. If $p(x, t)$ is the prediction of future reward [10] for state x at time t,

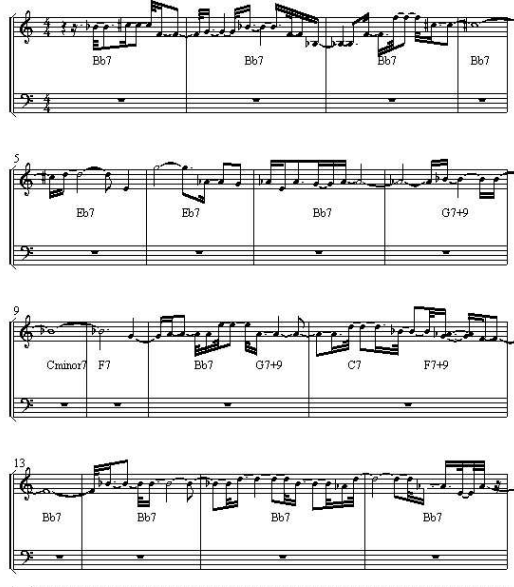

Figure 3: Phase1 trading 4 bars: 4 human, 4 machine, 4 human, 4 machine

$r_c(t+1) = r(t) + \gamma_c * p(x(t+1), t) - p(x(t), t)$ for $0 < \gamma_c < 1$.
The critic is a linear function of its inputs: $p(x(t), t) = \sum_{i=0}^{m} w_i(t)x_i(t)$ The weights are updated incrementally using the value of $r_c$:
$w_i(t+1) = w_i(t) + \beta r_c \partial p / \partial w_i$
$r_c$ is in effect an error signal, a difference between consecutive predicted rewards [10]. The critic also uses eligibility traces of the inputs, so $x(t)$ is actually $\bar{x}(t) = x(t) + e_x(t-1)$ where $e_x(t) = e_r \bar{x}(t)$. While this is all experimental, initial results show that the system with both the self-scalar and the critic performs better than with just one or without either one. A more systematic study is planned.

## 4   Results and Comments

Recall the rules of Section 1.2. Rules 1-4 are based on discussions with John Payne, a professional jazz musician and instructor of 25 years[1]. The rules by no means encompass all of jazz theory or practice but are a starting point. The notes in rule 2 were cast as good in a "hip" situation. The notion of hip requires human sophistication so for now these notes are reinforced if played sporadically on the dominant seventh. Rule 5 was added to discourage not playing any notes. Rules 5 and 6 focus on not allowing an output of one note for too long. Each chord is assigned a scale for rule 1. $\sigma_j$ is limited, $0.075 < \sigma_j < 0.2$, providing stability, and deliberate action uncertainty so different notes are played, for the same network state. Generally the goal of reinforcement learning is to find the best action for a given state, with uncertainty used for further exploration. Here, reinforcement learning finds the best set of actions for a given state. In a typical example using the phase 1 network prior to phase 2 improvement, the average reinforcement value according to the rule set is -.37 (on a scale from -1 to 1). After Phase 2, the average reinforcement value is .28 after 30-100 off-line presentations of the human solo of 1800 note events.

Figure 6 shows 12 bars of a human solo and 12 bars of a machine solo. The note durations

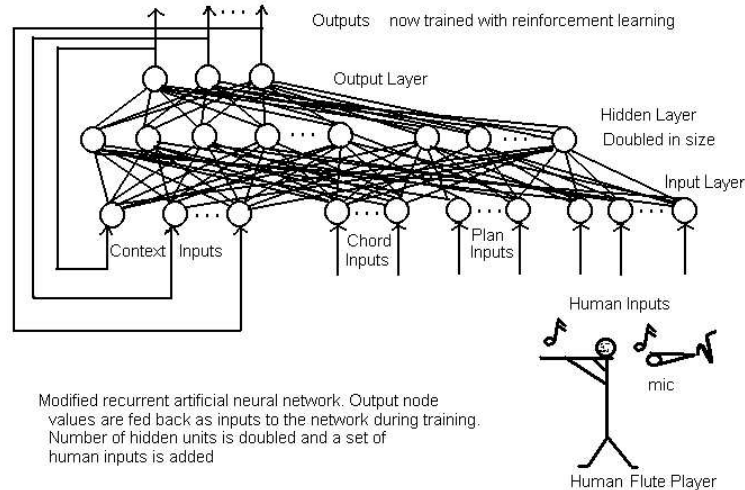

Figure 4: Recurrent reinforcement learning network with human input used in phase 2.

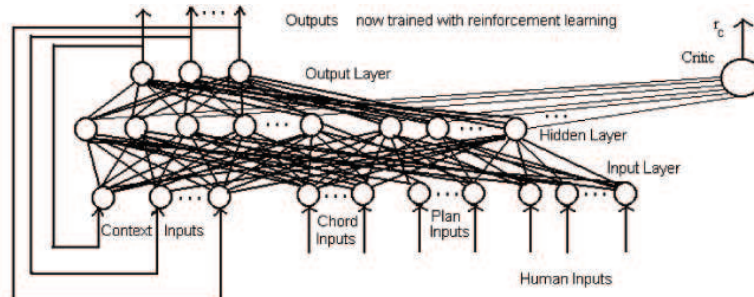

Figure 5: Phase 2 network with critic "piggy-backing" on hidden layer.

are shortened, reflecting the rules to prevent settling onto one note. The machine plays chord tones, such as Bb and D in bars 1 and 2. The high G is the 13 of Bb7, a hip note. In bars 3 and 4 it plays C sharp, a hip note (the #9 of Bb7) and high G. These notes are played in bars 9 and 11 on Bb7. In bars 5 and 6 the 9 and 13 (F and C) are played on Eb7. The natural 7 (D) reflects its heavy use in Rollins' melodies. Hip notes show up in bar 9 on Cm7: the 13 (G) and the 9 (D). In bar 11 G is played on G7+9 as is the hip flat 9 (the Ab). In bar 12, the Eb (flat 7 chord tone) is played on the F7+9. In bars 2, 4, 7, 9, and 10 the machine starts at the G at the top of the staff and descends through several chord tones, producing a recurring motif, an artifact of a "good" jazz solo. The phase 2 network has been used to interact with a human in real time while still learning. It keeps its recurrence since the human has a separate set of inputs.

A limitation to be addressed for CHIME is to move beyond one chord at a time. To achieve this, it must use more context, over more time. There are plenty of improvisation rules for chord progressions [8]. Because CHIME employs reinforcement learning, it has a stochastic element that allows it to play "outside the chord changes." A research topic is to understand how to enable it to do this more pointedly.

Figure 6: At left, 12 bars of human solo. At right, 12 bars the machine plays in response.

## Footnotes

[1]The rules are not meant to represent John Payne

## References

[1] J. Aebersold. *You can play Sonny Rollins. A New Approach to Jazz Improvisation Vol 8.* Jamey Aebersold, New Albany, IND., 1976.

[2] A. G. Barto, R. S. Sutton, and C. W. Anderson. Neuronlike adaptive element that can solve difficult learning control problems. *IEEE Transactions on Systems, Man, and Cybernetics*, SMC-13:834–846, 1983.

[3] H. Benbrahim and J. Franklin. Biped walking using reinforcement learning. *Robotics and Autonomous Systems*, 22:283–302, 1997.

[4] N. Griffith and P. Todd. *Musical Networks: Parallel Distributed Perception and Performance.* MIT Press, Cambridge MA, 1999.

[5] V. Gullapalli, J. Franklin, and H. Benbrahim. Acquiring robot skills via reinforcement learning. *IEEE Control Systems Magazine*, 1994.

[6] M. Jordan. Attractor dynamics and parallelism in a connectionist sequential machine. In *Proceedings of the Eighth Annual Conference of the Cognitive Science Society*, 1986.

[7] P. Messick. *Maximum MIDI.* Manning Publications, Greenwich, CT, 1988.

[8] S. Reeves. *Creative Jazz Improvisation. 2nd Ed.* Prentice Hall, Upper Saddle River NJ, 1995.

[9] M. A. Sabatella. *Whole Approach to Jazz Improvisation.* A.D.G. Productions, Lawndale CA, 1996.

[10] R. Sutton. Learning to predict by the methods of temporal differences. *Machine Learning*, 3:9–44, 1988.

[11] P. M. Todd. A connectionist approach to algorithmic composition. In P. M. Todd and e. D. Loy, editors, *Music and Connectionism*. MIT Press, Cambridge MA, 1991.

[12] P. M. Todd and e. D. Loy. *Music and Connectionism.* MIT Press, Cambridge, MA, 1991.

[13] R. J. Williams. Simple statistical gradient-following algorithms for connectionist reinforcement learning. *Machine Learning*, 8:229–256, 1992.
